# Breaking SVM Complexity
# with Cross-Training

**Gökhan H. Bakır**
Max Planck Institute
for Biological Cybernetics,
Tübingen, Germany
gb@tuebingen.mpg.de

**Léon Bottou**
NEC Labs America
Princeton NJ, USA
leon@bottou.org

**Jason Weston**
NEC Labs America
Princeton NJ, USA
jasonw@nec-labs.com

## Abstract

We propose to selectively remove examples from the training set using probabilistic estimates related to editing algorithms (Devijver and Kittler, 1982). This heuristic procedure aims at creating a separable distribution of training examples with minimal impact on the position of the decision boundary. It breaks the linear dependency between the number of SVs and the number of training examples, and sharply reduces the complexity of SVMs during both the training and prediction stages.

## 1 Introduction

The number of Support Vectors (SVs) has a dramatic impact on the efficiency of Support Vector Machines (Vapnik, 1995) during both the learning and prediction stages. Recent results (Steinwart, 2004) indicate that the number $k$ of SVs increases linearly with the number $n$ of training examples. More specifically,

$$k/n \longrightarrow 2\mathcal{B}_K \tag{1}$$

where $n$ is the number of training examples and $\mathcal{B}_K$ is the smallest classification error achievable with the SVM kernel $K$. When using a universal kernel such as the Radial Basis Function kernel, $\mathcal{B}_K$ is the Bayes risk $\mathcal{B}$, i.e. the smallest classification error achievable with any decision function.

The computational requirements of modern SVM training algorithms (Joachims, 1999; Chang and Lin, 2001) are very largely determined by the amount of memory required to store the active segment of the kernel matrix. When this amount exceeds the available memory, the training time increases quickly because some kernel matrix coefficients must be recomputed multiple times. During the final phase of the training process, the active segment always contains all the $k(k+1)/2$ dot products between SVs. Steinwart's result (1) then suggests that the critical amount of memory scales at least like $\mathcal{B}^2 n^2$. This can be practically prohibitive for problems with either big training sets or large Bayes risk (noisy problems). Large numbers of SVs also penalize SVMs during the prediction stage as the computation of the decision function requires a time proportional to the number of SVs.

When the problem is separable, i.e. $\mathcal{B} = 0$, equation (1) suggests[1] that the number $k$ of SVs increases less than linearly with the number $n$ of examples. This improves the scaling laws for the SVM computational requirements.

In this paper, we propose to selectively remove examples from the training set using probabilistic estimates inspired by training set editing algorithms (Devijver and Kittler, 1982). The removal procedure aims at creating a separable set of training examples without modifying the location of the decision boundary. Making the problem separable breaks the linear dependency between the number of SVs and the number of training examples.

## 2 Related work

### 2.1 Salient facts about SVMs

We focus now on the $C$-SVM applied to the two-class pattern recognition problem. See (Burges, 1998) for a concise reference. Given $n$ training patterns $x_i$ and their associated classes $y_i = \pm 1$, the SVM decision function is:

$$f(x) = \sum_{i=1}^{n} \alpha_i^* y_i K(x_i, x) + b^* \tag{2}$$

The coefficient $\alpha_i^*$ in (2) are obtained by solving a quadratic programing problem:

$$\alpha^* = \arg\max_{\alpha} \sum_i \alpha_i - \frac{1}{2} \sum_{i,j} \alpha_i \alpha_j y_i y_j K(x_i, x_j) \tag{3}$$

$$\text{subject to } \forall i,\ 0 \leq \alpha_i \leq C \text{ and } \sum_i \alpha_i y_i = 0$$

This optimization yields three categories of training examples depending on $\alpha_i^*$. Within each category, the possible values of the margins $y_i f(x_i)$ are prescribed by the Karush-Kuhn-Tucker optimality conditions.

- Examples such that $\alpha_i^* = C$ are called *bouncing SVs* or *margin errors* and satisfy $y_i f(x_i) < 1$. The set of bouncing SVs includes all training examples misclassified by the SVM, i.e. those which have a negative margin $y_i f(x_i) < 0$.
- Examples such that $0 < \alpha_i^* < C$ are called *ordinary SVs* and satisfy $y_i f(x_i) = 1$.
- Examples such that $\alpha_i^* = 0$ satisfy relation $y_i f(x_i) > 1$. These examples play no role in the SVM decision function (2). Retraining after discarding these examples would still yield the same SVM decision function (2).

These facts provide some insight into Steinwart's result (1). The SVM decision function, like any other decision rule, must asymptotically misclassify at least $\mathcal{B}n$ examples, where $\mathcal{B}$ is the Bayes risk. All these examples must therefore become bouncing SVs.

To illustrate dependence on the Bayes risk, we perform a linear classification task in two dimensions under varying amount of class overlap. The class distributions were uniform on a unit square with centers $c_1$ and $c_2$. Varying the distance between $c_1$ and $c_2$ allows us to control the Bayes risk. The results are shown in figure 1.

### 2.2 A posteriori reduction of the number of SVs.

Several techniques aim to reduce the prediction complexity of SVMs by expressing the SVM solution (2) with a smaller kernel expansion. Since one must compute the SVM solution before applying these post-processing techniques, they are not suitable for reducing the complexity of the training stage.

**Reduced Set Construction.** Burges (Burges, 1996) proposes to construct new patterns $z_j$ in order to define a compact approximation of the decision function (2). Reduced set construction usually involves solving a non convex optimization problem and is not applicable on arbitrary inputs such as graphs or strings.

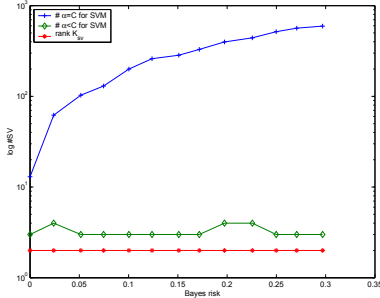

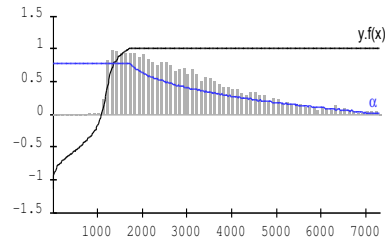

Figure 1: Effect of noise on the number of support vectors. The number of ordinary SVs stays almost constant whereas the number of bouncing SVs grows. Additional support vectors do not give extra information as indicated by the rank of the kernel matrix. See section 2.1.

Figure 2: Histogram of SVs selected by the $\ell_1$ penalization method on the MNIST 3-8 discrimination task. The initial SVs have been ordered on the $x$-axis by increasing margin $yf(x)$ and decreasing $\alpha$. See last paragraph in section 2.2.

**Reduced Set Selection.** The set of basis[2] functions $K(x_i, \cdot)$ associated with the SVs $x_i$ do not necessarily constitute a linearly independent family. The same decision function $f(\cdot)$ can then be expressed by multiple linear combination of the functions $K(x_i, \cdot)$. Reduced set selection methods attempt to select a subset of the SVs that is sufficient to express the SVM decision function. For instance, (Downs, Gates and Masters, 2001) propose to compute the row echelon form of the kernel matrix and discard SVs that lead to zero rows. This approach maintains the original SVM decision function.

In contrast, the $\ell_1$ penalization method suggested in (Schölkopf and Smola, 2002, sect. 18.4.2) simply attempts to construct a sufficiently good approximation of the original SVM decision function by solving

$$\arg\min_{\beta} \left\| \sum_i \alpha_i^* y_i K(x_i, \cdot) - \sum_i \beta_i y_i K(x_i, \cdot) \right\|_{\mathcal{K}}^2 + \lambda \sum_i |\beta_i| \tag{4}$$

where parameter $\lambda$ trades accuracy versus sparsity, and $\|\cdot\|_{\mathcal{K}}$ denotes the Reproducing Kernel Hilbert Space norm (Schölkopf and Smola, 2002, definition 2.9). Simplifying expression (4) yields a numerically tractable quadratic programming problem.

**Which examples are selected?** We have investigated the $\ell_1$ penalization method (4) as follows. We train a first SVM to discriminate digits 3 and 8 on the MNIST dataset (see section 4.2) after randomly swapping 10% of the class labels in the training set. We then select a subset of the resulting support vectors using the $\ell_1$ penalization method.

Choosing $\lambda$ is quite difficult in practice. To evaluate the accuracy of the procedure, we train a second SVM on the selected vectors, compare its recognition accuracy with that of the first SVM. This was best achieved by enforcing the constraint $\beta_i \geq 0$ in (4) because the second SVM cannot return an expansion with negative coefficients.

Figure 2 shows the histogram of selected SVs. The initial support vectors have been ordered on the $x$-axis by increasing values of $y_i f(x_i)$, and, in the case of margin SVs, by decreasing values of $\alpha_i$. The selected SVs includes *virtually no misclassified SVs*, but instead concentrates on *SVs with large $\alpha_i$*.

This result suggests that simple pre-processing methods might indicate which training examples are really critical for SVM classification.

## 2.3 Training set editing techniques

We now consider techniques for reducing the set of training examples before running a training algorithm. Reducing the amount of training data is indeed an obvious way to reduce the complexity of training. Quantization and clustering methods might be used to achieve this goal. These methods however reduce the training data without considering the loss function of interest, and therefore sacrifice classification accuracy. We focus instead on *editing techniques*, i.e. techniques for discarding selected training examples with the aim of achieving similar or better classification accuracy.

Two prototypical editing techniques, MULTIEDIT and CONDENSE, have been thoroughly studied (Devijver and Kittler, 1982, chapter 3) in the context of the nearest neighbor (1-NN) classification rule.

**Removing interior examples.** The CONDENSE algorithm was first described by (Hart, 1968). This algorithm selects a subset of the training examples whose 1-NN decision boundary still classifies correctly all of the initial training examples:

**Algorithm 1 (CONDENSE).**

    *1 Select a random training example and put it in set $R$.*

    *2 For each training example $i = 1, \ldots, n$ : classify example $i$ using the 1-NN rule with set $R$ as the training set, and insert it into $R$ if it is misclassified.*

    *3 Return to step 2 if $R$ has been modified during the last pass.*

    *4 The final contents of $R$ constitute the condensed training set.*

This is best understood when both classes form homogeneous clusters in the feature space. Algorithm 1 discards training examples located in the interior of each cluster.

This strategy works poorly when there is a large overlap between the pattern distributions of both classes, that is to say when the Bayes risk $\mathcal{B}$ is large. Consider for instance a feature space region where $P(y = +1 \mid x) > P(y = -1 \mid x) > 0$. A small number of training examples of class $y = -1$ can still appear in such a region. We say that they are located on the wrong side of the Bayes decision boundary. Asymptotically, all such training examples belong to the condensed training set in order to ensure that they are properly recognized as members of class $y = -1$.

**Removing noise examples.** The *Edited Nearest Neighbor* rule (Wilson, 1972) suggests to first discard all training examples that are misclassified when applying the 1-NN rule using all $n - 1$ remaining examples as the training set. It was shown that removing these examples *improves* the asymptotic performance of the nearest neighbor rule. Whereas the 1-NN risk is asymptotically bounded by $2\mathcal{B}$, the Edited 1-NN risk is asymptotically bounded by $1.2\,\mathcal{B}$, where $\mathcal{B}$ is the Bayes risk.

The MULTIEDIT algorithm (Devijver and Kittler, 1982, section 3.11) asymptotically discards *all* the training examples located on the wrong side of the Bayes decision boundary. The asymptotic risk of the multi-edited nearest neighbor rule is the Bayes risk $\mathcal{B}$.

**Algorithm 2 (MULTIEDIT).**

    *1 Divide randomly the training data into $s$ splits $\mathcal{S}_1, \ldots, \mathcal{S}_s$. Let us call $f_i$ the 1-NN classifier that uses $\mathcal{S}_i$ as the training set.*

    *2 Classify all examples in $\mathcal{S}_i$ using the classifier $f_{(i+1) \bmod s}$ and discard all misclassified examples.*

    *3 Gather all the remaining examples and return to step 1 if any example has been discarded during the last $T$ iterations.*

    *4 The remaining examples constitute the multiedited training set.*

By discarding examples located on the wrong side of the Bayes decision boundary, algorithm MULTIEDIT constructs a new training set whose apparent distribution has the same Bayes decision boundary as the original problem, but with Bayes risk equal to $0$. Devijver and Kittler claim that MULTIEDIT produces an ideal training set for CONDENSE.

Algorithm MULTIEDIT also discards some proportion of training examples located on the correct side of Bayes decision boundary. Asymptotically this does not matter. However this is often a problem in practice...

## 2.4 Editing algorithms and SVMs

Training examples recognized with high confidence usually do not appear in the SVM solution (2) because they do not become support vectors. On the other hand, outliers always become support vectors. Intuitively, SVMs display the properties of the CONDENSE but lack the properties of the MULTIEDIT algorithm.

The mathematical proofs for the asymptotic properties of MULTIEDIT depend on the specific nature of the 1-NN classification rule. The MULTIEDIT algorithm itself could be identically defined for any classifier. This suggests (but does not prove) that these properties might remain valid for SVM classifiers[3].

> This contribution is an *empirical* attempt to endow Support Vector Machines with the properties of the MULTIEDIT algorithm.

Editing SVM training sets implicitly modifies the SVM loss function in a way that relates to robust statistics. Editing alters the apparent distribution of training examples such that the class distributions $P(x \mid y = 1)$ and $P(x \mid y = -1)$ no longer overlap. If the class distributions were known, this could be done by trimming the tails of the class distributions. A similar effect could be obtained by altering the SVM loss function (the hinge loss) into a non convex loss function that gives less weight to outliers.

# 3 Cross-Training

Cross-Training is a representative algorithm of such combinations of SVMs and editing algorithms. It begins with creating $s$ subsets of the training set with $r$ examples each. Independent SVMs are then trained on each subset. The decision functions of these SVMs are then used to discard two types of training examples: those which are confidently recognized, as in CONDENSE, and those which are misclassified, as in MULTIEDIT. A final SVM is then trained using the remaining examples.

**Algorithm 3 (CROSSTRAINING).**

*1 Create $s$ subsets of size $r$ by randomly drawing $r/2$ examples of each class.*

*2 Train $s$ independent SVMs $f_1, \ldots, f_s$ using each of the subsets as the training set.*

*3 For each training example $(x_i, y_i)$ estimate the margin average $m_i$ and variance $v_i$:*

$$m_i = \tfrac{1}{s} \sum_{r=1}^{s} y_i f_r(x_i) \qquad v_i = \tfrac{1}{s} \sum_{r=1}^{s} (m_i - y_i f_r(x_i))^2$$

*4 Discard all training examples for which $m_i + v_i < 0$.*

*5 Discard all training examples for which $m_i - v_i > 1$.*

*6 Train a final SVM on the remaining training examples.*

The apparent simplicity of this algorithm hides a lot of hyperparameters. The value of the C parameters for the SVMs at steps [2] and [6] has a considerable effect on the overall performance of the algorithm.

For the first stage SVMs, we choose the $C$ parameter which yields the best performance on training sets of size $r$. For the second stage SVMs, we choose the $C$ parameter which yields the best overall performance measured on a separate validation set.

Furthermore, we discovered that the discarding steps tend to produce a final set of training examples with very different numbers of examples for each class. Specific measures to alleviate this problem are discussed in section 4.3.

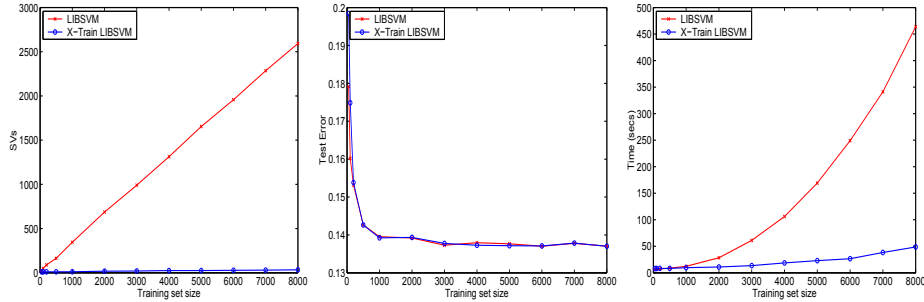

Figure 3: Comparing LIBSVM and Cross-Training on a toy problem of two Gaussian clouds for increasing number of training points. Cross-Training gives an almost constant number of support vectors (left figure) for increasing training set size, whereas in LIBSVM the number of support vectors increases linearly. The error rates behave similarly (middle figure), and Cross-Training gives an improved training time (right figure). See section 4.1.

# 4 Experiments

## 4.1 Artificial Data

We first constructed artificial data, by generating two classes from two Gaussian clouds in 10 dimensions with means $(1, 1, 1, 1, 1, 0, 0, 0, 0, 0)$ and $(-1, -1, -1, -1, -1, 0, 0, 0, 0, 0)$ and standard deviation 4. We trained a linear SVM for differing amounts of training points, selecting $C$ via cross validation. We compare the performance of LIBSVM[4] with Cross-Training using LIBSVM with $s = 5$, averaging over 10 splits. The results given in figure 3 show a reduction in SVs and computation time using Cross-Training, with no loss in accuracy.

## 4.2 Artificial Noise

Our second experiment involves the discrimination of digits 3 and 8 in the MNIST[5] database. Artificial noise was introduced by swapping the labels of 0%, 5%, 10% and 15% of the examples. There are 11982 training examples and 1984 testing examples. All experiments were carried out using LIBSVM's $\nu$-SVM (Chang and Lin, 2001) with the RBF kernel ($\gamma = 0.005$). Cross-Training was carried out by splitting the 11982 training examples into 5 subsets. Figure 4 reports our results for various amounts of label noise. The number of SVs (left figure) increases linearly for the standard SVM and stays constant for the Cross-Training SVM. The test errors (middle figure) seem similar. Since our label noise is artificial, we can also measure the misclassification rate on the unmodified testing set (right figure). This measurement shows a slight loss of accuracy without statistical significance.

## 4.3 Benchmark Data

Finally the cross-training algorithm was applied to real data sets from both the ANU repository[6] and from the the UCI repository[7].

Experimental results were quite disappointing until we realized that the discarding steps tends to produce training sets with very different numbers of examples for each class. To alleviate this problem, after training each SVM, we choose the value of the threshold $b^*$

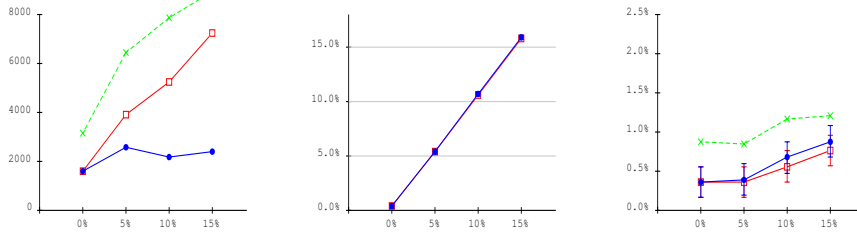

Figure 4: Number of SVs (left figure) and test error (middle figure) for varying amounts of label noise on the MNIST 3-8 discrimination task. The $x$-axis in all graphs shows the amount of label noise; white squares correspond to LIBSVM; black circles to Cross-Training; dashed lines to bagging the first stage Cross-Training SVMs. The last graph (right figure) shows the test error measured without label noise. See section 4.2

in (2) which achieves the best validation performance. We also attempt to balance the final training set by re-inserting examples discarded during step [5] of the cross-training algorithm.

Experiments were carried out using RBF kernels with the kernel width reported in the literature. In the SVM experiments, the value of parameter $C$ was determined by cross-validation and then used for training a SVM on the full dataset. In the cross-training experiments, we make a validation set by taking $r/3$ examples from the training set. These examples are only used for choosing the values of $C$ and for adjusting the SVM thresholds. Details and source code are available[8].

| Dataset | Train Size | Test Size | SVM Perf.[%] | SVM #SV | XTrain Subsets | XTrain Perf.[%] | XTrain #SV |
|---|---|---|---|---|---|---|---|
| Banana | 400 | 4900 | **89.0** | 111 | 5×200 | 88.2 | **51** |
| Waveform | 400 | 4600 | **90.2** | 172 | 5×200 | 88.7 | **87** |
| Splice | 1000 | 2175 | 90.0 | 601 | 5×300 | 89.9 | **522** |
| Adult | 3185 | 16280 | 84.2 | 1207 | 5×700 | 84.2 | **606** |
| Adult | 32560 | 16280 | 85.1 | 11325 | 5×6000 | 84.8 | **1194** |
| Forest | 50000 | 58100 | **90.3** | 12476 | 5×10000 | 89.2 | **7967** |
| Forest | 90000 | 58100 | **91.6** | 18983 | 5×18000 | 90.7 | **13023** |
| Forest | 200000 | 58100 | — | — | 8×30000 | **92.1** | **19526** |

Table 1: Comparison of SVM and Cross-Training results on standard benchmark data sets.

The columns in table 1 contain the dataset name, the size of the training set used for the experiment, the size of the test set, the SVM accuracy and number of SVs, the Cross-Training subset configuration, accuracy, and final number of SVs. Bold typeface indicates which differences were statistically significant according to a paired test. These numbers should be considered carefully because they are impacted by the discrete nature of the grid search for parameter $C$. The general trend still indicates that Cross-Training causes a slight loss of accuracy but requires much less SVs.

Our largest training set contains 200000 examples. Training a standard SVM on such a set takes about one week of computation. We do not report this result because it was not practical to determine a good value of $C$ for this experiment. Cross-Training with specified hyperparameters runs overnight. Cross-Training with hyperparameter grid searches runs in two days.

We do not report detailed timing results because much of the actual time can be attributed

to the search for the proper hyperparameters. Timing results would then depend on loosely controlled details of the hyperparameter grid search algorithms.

## 5 Discussion

We have suggested to combine SVMs and training set editing techniques to break the linear relationship between number of support vectors and number of examples. Such combinations raise interesting theoretical questions regarding the relative value of each of the training examples.

Experiments with a representative algorithm, namely Cross-Training, confirm that both the training and the recognition time are sharply reduced. On the other hand, Cross-Training causes a minor loss of accuracy, comparable to that of reduced set methods (Burges, 1996), and seems to be more sensitive than SVMs in terms of parameter tuning.

Despite these drawbacks, Cross-Training provides a practical means to construct kernel classifiers with significantly larger training sets.

## 6 Acknowledgement

We thank Hans Peter Graf, Eric Cosatto and Vladimir Vapnik for their advice and support. Part of this work was funded by NSF grant CCR-0325463.

**References**

Burges, C. J. C. (1996). Simplified Support Vector Decision Rules. In Saitta, L., editor, *Proceedings of the 13th International Conference on Machine Learning*, pages 71–77, San Mateo, CA. Morgan Kaufmann.

Burges, C. J. C. (1998). A Tutorial on Support Vector Machines for Pattern Recognition. *Data Mining and Knowledge Discovery*, 2(2):121–167.

Chang, C.-C. and Lin, C.-J. (2001). Training $\nu$-Support Vector Classifiers: Theory and Algorithms. *Neural Computation*, 13(9):2119–2147.

Devijver, P. and Kittler, J. (1982). *Pattern Recogniton, A statistical approach*. Prentice Hall, Englewood Cliffs.

Downs, T., Gates, K. E., and Masters, A. (2001). Exact Simplification of Support Vector Solutions. *Journal of Machine Learning Research*, 2:293–297.

Hart, P. (1968). The condensed nearest neighbor rule. *IEEE Transasctions on Information Theory*, 14:515–516.

Joachims, T. (1999). Making Large–Scale SVM Learning Practical. In Schölkopf, B., Burges, C. J. C., and Smola, A. J., editors, *Advances in Kernel Methods — Support Vector Learning*, pages 169–184, Cambridge, MA. MIT Press.

Schölkopf, B. and Smola, A. J. (2002). *Learning with Kernels*. MIT Press, Cambridge, MA.

Steinwart, I. (2004). Sparseness of Support Vector Machines—Some Asymptotically Sharp Bounds. In Thrun, S., Saul, L., and Schölkopf, B., editors, *Advances in Neural Information Processing Systems 16*. MIT Press, Cambridge, MA.

Vapnik, V. N. (1995). *The Nature of Statistical Learning Theory*. Springer Verlag, New York.

Wilson, D. L. (1972). Asymptotic properties of the nearest neighbor rules using edited data. *IEEE Transactions on Systems, Man, and Cybernetics*, 2:408–420.

## Footnotes

[1] See also (Steinwart, 2004, remark 3.8)

[2]We use the customary name *basis functions* despite linear dependence. . .

[3]Further comfort comes from the knowledge that a SVM with the RBF kernel and without threshold term $b$ implements the 1-NN rule when the RBF radius tends to zero.

[4] http://www.csie.ntu.edu.tw/∼cjlin/libsvm/

[5] http://yann.lecun.com/exdb/mnist

[6] http://mlg.anu.edu.au/∼raetsch/data/index.html

[7] ftp://ftp.ics.uci.edu/pub/machine-learning-databases

[8] http://www.kyb.tuebingen.mpg.de/bs/people/gb/xtraining
